# Optimal learning rates for least squares SVMs using Gaussian kernels

**M. Eberts, I. Steinwart**
Institute for Stochastics and Applications
University of Stuttgart
D-70569 Stuttgart
{eberts,ingo.steinwart}@mathematik.uni-stuttgart.de

## Abstract

We prove a new oracle inequality for support vector machines with Gaussian RBF kernels solving the regularized least squares regression problem. To this end, we apply the modulus of smoothness. With the help of the new oracle inequality we then derive learning rates that can also be achieved by a simple data-dependent parameter selection method. Finally, it turns out that our learning rates are asymptotically optimal for regression functions satisfying certain standard smoothness conditions.

## 1 Introduction

On the basis of i.i.d. observations $D := ((x_1, y_1), \ldots, (x_n, y_n))$ of input/output observations drawn from an unknown distribution P on $X \times Y$, where $Y \subset \mathbb{R}$, the goal of non-parametric least squares regression is to find a function $f_D : X \to \mathbb{R}$ such that, for the least squares loss $L : Y \times \mathbb{R} \to [0, \infty)$ defined by $L(y, t) = (y - t)^2$, the risk

$$\mathcal{R}_{L,\mathrm{P}}(f_D) := \int_{X \times Y} L(y, f_D(x)) \, d\mathrm{P}(x, y) = \int_{X \times Y} (y - f_D(x))^2 \, d\mathrm{P}(x, y)$$

is small. This means $\mathcal{R}_{L,\mathrm{P}}(f_D)$ has to be close to the optimal risk

$$\mathcal{R}_{L,\mathrm{P}}^* := \inf \{\mathcal{R}_{L,\mathrm{P}}(f) \mid f : X \to \mathbb{R} \text{ measureable}\} \,,$$

called the Bayes risk with respect to P and $L$. It is well known that the function $f_{L,\mathrm{P}}^* : X \to \mathbb{R}$ defined by $f_{L,\mathrm{P}}^*(x) = \mathbb{E}_{\mathrm{P}}(Y|x)$, $x \in X$, is the only function for which the Bayes risk is attained. Furthermore, some simple transformations show

$$\mathcal{R}_{L,\mathrm{P}}(f) - \mathcal{R}_{L,\mathrm{P}}^* = \int_X |f - f_{L,\mathrm{P}}^*|^2 \, d\mathrm{P}_X = \|f - f_{L,\mathrm{P}}^*\|_{L_2(\mathrm{P}_X)}^2 \,, \tag{1}$$

where $\mathrm{P}_X$ is the marginal distribution of $P$ on $X$.

In this paper, we assume that $X \subset \mathbb{R}^d$ is a non-empty, open and bounded set such that its boundary $\partial X$ has Lebesgue measure 0, $Y := [-M, M]$ for some $M > 0$ and P is a probability measure on $X \times Y$ such that $\mathrm{P}_X$ is the uniform distribution on $X$. In Section 2 we also discuss that this condition can easily be generalized by assuming that $\mathrm{P}_X$ on $X$ is absolutely continuous with respect to the Lebesgue measure on $X$ such that the corresponding density of $P_X$ is bounded away from 0 and $\infty$. Recall that because of the first assumption, it suffices to restrict considerations to decision functions $f : X \to [-M, M]$. To be more precise, if, we denote the clipped value of some $t \in \mathbb{R}$ by $\widehat{t}$, that is

$$\widehat{t} := \begin{cases} -M & \text{if } t < -M \\ t & \text{if } t \in [-M, M] \\ M & \text{if } t > M \,, \end{cases}$$

then it is easy to check that

$$\mathcal{R}_{L,\mathrm{P}}(\widehat{f}) \leq \mathcal{R}_{L,\mathrm{P}}(f) \ ,$$

for all $f : X \to \mathbb{R}$.

The non-parametric least squares problem can be solved in many ways. Several of them are e.g. described in [1]. In this paper, we use SVMs to find a solution for the non-parametric least squares problem by solving the regularized problem

$$f_{\mathrm{D},\lambda} = \arg \min_{f \in H} \lambda \|f\|_H^2 + \mathcal{R}_{L,\mathrm{D}}(f) \ . \tag{2}$$

Here, $\lambda > 0$ is a fixed real number, $H$ is a reproducing kernel Hilbert space (RKHS) over $X$, and $\mathcal{R}_{L,\mathrm{D}}(f)$ is the empirical risk of $f$, that is

$$\mathcal{R}_{L,\mathrm{D}}(f) = \frac{1}{n} \sum_{i=1}^{n} L\left(y_i, f\left(x_i\right)\right) \ .$$

In this work we restrict our considerations to Gaussian RBF kernels $k_\gamma$ on $X$, which are defined by

$$k_\gamma\left(x, x'\right) = \exp\left(-\frac{\|x - x'\|_2^2}{\gamma^2}\right) \ , \qquad x, x' \in X \ ,$$

for some width $\gamma \in (0, 1]$. Our goal is to deduce asymptotically optimal learning rates for the SVMs (2) using the RKHS $H_\gamma$ of $k_\gamma$. To this end, we first establish a general oracle inequality. Based on this oracle inequality, we then derive learning rates if the regression function is contained in some Besov space. It will turn out, that these learning rates are asymptotically optimal. Finally, we show that these rates can be achieved by a simple data-dependent parameter selection method based on a hold-out set.

The rest of this paper is organized as follows: The next section presents the main theorems and as a consequence of these theorems some corollaries inducing asymptotically optimal learning rates for regression functions contained in Sobolev or Besov spaces. Section 3 states some, for the proof of the main statement necessary, lemmata and a version of [2, Theorem 7.23] applied to our special case as well as the proof of the main theorem. Some further proofs and additional technical results can be found in the appendix.

## 2 Results

In this section we present our main results including the optimal rates for LS-SVMs using Gaussian kernels. To this end, we first need to introduce some function spaces, which are later assumed to contain the regression function.

Let us begin by recalling from, e.g. [3, p. 44], [4, p. 398], and [5, p. 360], the modulus of smoothness:

**Definition 1.** *Let $\Omega \subset \mathbb{R}^d$ with non-empty interior, $\nu$ be an arbitrary measure on $\Omega$, and $f : \Omega \to \mathbb{R}^d$ be a function with $f \in L_p(\nu)$ for some $p \in (0, \infty)$. For $r \in \mathbb{N}$, the $r$-th modulus of smoothness of $f$ is defined by*

$$\omega_{r,L_p(\nu)}(f, t) = \sup_{\|h\|_2 \leq t} \|\triangle_h^r(f, \cdot)\|_{L_p(\nu)} \ , \qquad t \geq 0 \ ,$$

*where $\|\cdot\|_2$ denotes the Euclidean norm and the $r$-th difference $\triangle_h^r(f, \cdot)$ is defined by*

$$\triangle_h^r(f, x) = \begin{cases} \sum_{j=0}^r \binom{r}{j} (-1)^{r-j} f(x + jh) & \text{if } x \in \Omega_{r,h} \\ 0 & \text{if } x \notin \Omega_{r,h} \end{cases}$$

*for $h = (h_1, \ldots, h_d) \in \mathbb{R}^d$ with $h_i \geq 0$ and $\Omega_{r,h} := \{x \in \Omega : x + sh \in \Omega \ \forall s \in [0, r]\}$.*

It is well-known that the modulus of smoothness with respect to $L_p(\nu)$ is a nondecreasing function of $t$ and for the Lebesgue measure on $\Omega$ it satisfies

$$\omega_{r,L_p(\Omega)}(f, t) \leq \left(1 + \frac{t}{s}\right)^r \omega_{r,L_p(\Omega)}(f, s) \ , \tag{3}$$

for all $f \in L_p(\Omega)$ and all $s > 0$, see e.g. [6, (2.1)]. Moreover, the modulus of smoothness can be used to define the scale of Besov spaces. Namely, for $1 \leq p, q \leq \infty$, $\alpha > 0$, $r := \lfloor \alpha \rfloor + 1$, and an arbitrary measure $\nu$, the Besov space $B_{p,q}^{\alpha}(\nu)$ is

$$B_{p,q}^{\alpha}(\nu) := \left\{ f \in L_p(\nu) : |f|_{B_{p,q}^{\alpha}(\nu)} < \infty \right\} ,$$

where, for $1 \leq q < \infty$, the seminorm $|\cdot|_{B_{p,q}^{\alpha}(\nu)}$ is defined by

$$|f|_{B_{p,q}^{\alpha}(\nu)} := \left( \int_0^{\infty} \left( t^{-\alpha} \omega_{r, L_p(\nu)}(f, t) \right)^q \frac{dt}{t} \right)^{\frac{1}{q}} ,$$

and, for $q = \infty$, it is defined by

$$|f|_{B_{p,\infty}^{\alpha}(\nu)} := \sup_{t > 0} \left( t^{-\alpha} \omega_{r, L_p(\nu)}(f, t) \right) .$$

In both cases the norm of $B_{p,q}^{\alpha}(\nu)$ can be defined by $\|f\|_{B_{p,q}^{\alpha}(\nu)} := \|f\|_{L_p(\nu)} + |f|_{B_{p,q}^{\alpha}(\nu)}$, see e.g. [3, pp. 54/55] and [4, p. 398]. Finally, for $q = \infty$, we often write $B_{p,\infty}^{\alpha}(\nu) = \text{Lip}^*(\alpha, L_p(\nu))$ and call $\text{Lip}^*(\alpha, L_p(\nu))$ the generalized Lipschitz space of order $\alpha$. In addition, it is well-known, see e.g. [7, p. 25 and p. 44], that the Sobolev spaces $W_p^{\alpha}(\mathbb{R}^d)$ fall into the scale of Besov spaces, namely

$$W_p^{\alpha}(\mathbb{R}^d) \subset B_{p,q}^{\alpha}(\mathbb{R}^d) \tag{4}$$

for $\alpha \in \mathbb{N}$, $p \in (1, \infty)$, and $\max\{p, 2\} \leq q \leq \infty$ and especially $W_2^{\alpha}(\mathbb{R}^d) = B_{2,2}^{\alpha}(\mathbb{R}^d)$.

For our results we need to extend functions $f : \Omega \to \mathbb{R}$ to functions $\hat{f} : \mathbb{R}^d \to \mathbb{R}$ such that the smoothness properties of $f$ described by some Sobolev or Besov space are preserved by $\hat{f}$. Recall that Stein's Extension Theorem guarantees the existence of such an extension, whenever $\Omega$ is a bounded Lipschitz domain. To be more precise, in this case there exists a linear operator $\mathfrak{E}$ mapping functions $f : \Omega \to \mathbb{R}$ to functions $\mathfrak{E}f : \mathbb{R}^d \to \mathbb{R}$ with the properties:

(a) $\mathfrak{E}(f)_{|\Omega} = f$, that is, $\mathfrak{E}$ is an extension operator.

(b) $\mathfrak{E}$ continuously maps $W_p^m(\Omega)$ into $W_p^m(\mathbb{R}^d)$ for all $p \in [1, \infty]$ and all integer $m \geq 0$. That is, there exist constants $a_{m,p} \geq 0$, such that, for every $f \in W_p^m(\Omega)$, we have

$$\|\mathfrak{E}f\|_{W_p^m(\mathbb{R}^d)} \leq a_{m,p} \|f\|_{W_p^m(\Omega)} . \tag{5}$$

(c) $\mathfrak{E}$ continuously maps $B_{p,q}^{\alpha}(\Omega)$ into $B_{p,q}^{\alpha}(\mathbb{R}^d)$ for all $p \in (1, \infty)$, $q \in (0, \infty]$ and all $\alpha > 0$. That is, there exist constants $a_{\alpha,p,q} \geq 0$, such that, for every $f \in B_{p,q}^{\alpha}(\Omega)$, we have

$$\|\mathfrak{E}f\|_{B_{p,q}^{\alpha}(\mathbb{R}^d)} \leq a_{\alpha,p,q} \|f\|_{B_{p,q}^{\alpha}(\Omega)} .$$

For detailed conditions on $\Omega$ ensuring the existence of $\mathfrak{E}$, we refer to [8, p. 181] and [9, p. 83]. Property (c) follows by some interpolation argument since $B_{p,q}^{\alpha}$ can be interpreted as interpolation space of the Sobolev spaces $W_p^{m_0}$ and $W_p^{m_1}$ for $q \in [1, \infty]$, $p \in (1, \infty)$, $\theta \in (0, 1)$ and $m_0, m_1 \in \mathbb{N}_0$ with $m_0 \neq m_1$ and $\alpha = m_0(1 - \theta) + m_1\theta$, see [10, pp. 65/66] for more details. In the following, we always assume that we do have such an extension operator $\mathfrak{E}$. Moreover, if $\mu$ is the Lebesgue measure on $\Omega$, such that $\partial\Omega$ has Lebesgue measure 0, the canonical extension of $\mu$ to $\mathbb{R}^d$ is given by $\widetilde{\mu}(A) := \mu(A \cap \Omega)$ for all measurable $A \subset \mathbb{R}^d$. However, in a slight abuse of notation, we often write $\mu$ instead of $\widetilde{\mu}$, since this simplifies the presentation. Analogously, we proceed for the uniform distribution on $\Omega$ and its canonical extension to $\mathbb{R}^d$ and the same convention will be applied to measures $P_X$ on $\Omega$ that are absolutely continuous w.r.t. the Lebesgue measure.

Finally, in order to state our main results, we denote the closed unit ball of the $d$-dimensional Euclidean space by $B_{\ell_2^d}$.

**Theorem 1.** *Let $X \subset B_{\ell_2^d}$ be a domain such that we have an extension operator $\mathfrak{E}$ in the above sense. Furthermore, let $M > 0$, $Y := [-M, M]$, and $P$ be a distribution on $X \times Y$ such that $P_X$ is the uniform distribution on $X$. Assume that we have fixed a version $f_{L,P}^*$ of the regression*

function such that $f_{L,\mathrm{P}}^*(x) = \mathbb{E}_\mathrm{P}(Y|x) \in [-M, M]$ for all $x \in X$. Assume that, for $\alpha \geq 1$ and $r := \lfloor \alpha \rfloor + 1$, there exists a constant $c > 0$ such that, for all $t \in (0, 1]$, we have

$$\omega_{r, L_2(\mathbb{R}^d)}\left(\mathfrak{E} f_{L,\mathrm{P}}^*, t\right) \leq ct^\alpha . \tag{6}$$

Then, for all $\varepsilon > 0$ and $p \in (0, 1)$ there exists a constant $K > 0$ such that for all $n \geq 1$, $\tau \geq 1$, and $\lambda > 0$, the SVM using the RKHS $H_\gamma$ satisfies

$$\lambda \|f_{\mathrm{D},\lambda}\|_{H_\gamma}^2 + \mathcal{R}_{L,\mathrm{P}}(\widehat{f}_{\mathrm{D},\lambda}) - \mathcal{R}_{L,\mathrm{P}}^* \leq K\lambda\gamma^{-d} + Kc^2\gamma^{2\alpha} + K\frac{\gamma^{-(1-p)(1+\varepsilon)d}}{\lambda^p n} + \frac{K\tau}{n}$$

with probability $\mathrm{P}^n$ not less than $1 - e^{-\tau}$.

With this oracle inequality we can derive learning rates for the learning method (2).

**Corollary 1.** *Under the assumptions of Theorem 1 and for $\varepsilon > 0$, $p \in (0, 1)$, and $\tau \geq 1$ fixed, we have, for all $n \geq 1$,*

$$\lambda_n \|f_{\mathrm{D},\lambda_n}\|_{H_{\gamma_n}}^2 + \mathcal{R}_{L,\mathrm{P}}(\widehat{f}_{\mathrm{D},\lambda_n}) - \mathcal{R}_{L,\mathrm{P}}^* \leq Cn^{-\frac{2\alpha}{2\alpha+2\alpha p + dp + (1-p)(1+\varepsilon)d}}$$

*with probability $\mathrm{P}^n$ not less than $1 - e^{-\tau}$ and with*

$$\lambda_n = c_1 n^{-\frac{2\alpha+d}{2\alpha+2\alpha p + dp + (1-p)(1+\varepsilon)d}} ,$$

$$\gamma_n = c_2 n^{-\frac{1}{2\alpha+2\alpha p + dp + (1-p)(1+\varepsilon)d}} .$$

*Here, $c_1 > 0$ and $c_2 > 0$ are user-specified constants and $C > 0$ is a constant independent of $n$.*

Note that for every $\rho > 0$ we can find $\varepsilon, p \in (0, 1)$ sufficiently close to 0 such that the learning rate in Corollary 1 is at least as fast as

$$n^{-\frac{2\alpha}{2\alpha+d}+\rho} .$$

To achieve these rates, however, we need to set $\lambda_n$ and $\gamma_n$ as in Corollary 1, which in turn requires us to know $\alpha$. Since in practice we usually do not know this value, we now show that a standard training/validation approach, see e.g. [2, Chapters 6.5, 7.4, 8.2], achieves the same rates adaptively, i.e. without knowing $\alpha$. To this end, let $\Lambda := (\Lambda_n)$ and $\Gamma := (\Gamma_n)$ be sequences of finite subsets $\Lambda_n, \Gamma_n \subset (0, 1]$. For a data set $D := ((x_1, y_1), \ldots, (x_n, y_n))$, we define

$$D_1 := ((x_1, y_1), \ldots, (x_m, y_m))$$
$$D_2 := ((x_{m+1}, y_{m+1}), \ldots, (x_n, y_n))$$

where $m := \lfloor \frac{n}{2} \rfloor + 1$ and $n \geq 4$. We will use $D_1$ as a training set by computing the SVM decision functions

$$f_{\mathrm{D}_1,\lambda,\gamma} := \arg\min_{f \in H_\gamma} \lambda \|f\|_{H_\gamma}^2 + \mathcal{R}_{L,\mathrm{D}_1}(f), \qquad (\lambda, \gamma) \in \Lambda_n \times \Gamma_n$$

and use $D_2$ to determine $(\lambda, \gamma)$ by choosing a $(\lambda_{\mathrm{D}_2}, \gamma_{\mathrm{D}_2}) \in \Lambda_n \times \Gamma_n$ such that

$$\mathcal{R}_{L,\mathrm{D}_2}\left(f_{\mathrm{D}_1,\lambda_{\mathrm{D}_2},\gamma_{\mathrm{D}_2}}\right) = \min_{(\lambda,\gamma)\in\Lambda_n\times\Gamma_n} \mathcal{R}_{L,\mathrm{D}_2}\left(f_{\mathrm{D}_1,\lambda,\gamma}\right) .$$

**Theorem 2.** *Under the assumptions of Theorem 1 we fix sequences $\Lambda := (\Lambda_n)$ and $\Gamma := (\Gamma_n)$ of finite subsets $\Lambda_n, \Gamma_n \subset (0, 1]$ such that $\Lambda_n$ is an $\epsilon_n$-net of $(0, 1]$ and $\Gamma_n$ is an $\delta_n$-net of $(0, 1]$ with $\epsilon_n \leq n^{-1}$ and $\delta_n \leq n^{-\frac{1}{2+d}}$. Furthermore, assume that the cardinalities $|\Lambda_n|$ and $|\Gamma_n|$ grow polynomially in $n$. Then, for all $\rho > 0$, the TV-SVM producing the decision functions $f_{\mathrm{D}_1,\lambda_{\mathrm{D}_2},\gamma_{\mathrm{D}_2}}$ learns with the rate*

$$n^{-\frac{2\alpha}{2\alpha+d}+\rho} \tag{7}$$

*with probability $\mathrm{P}^n$ not less than $1 - e^{-\tau}$.*

What is left to do is to relate Assumption (6) with the function spaces introduced earlier, such that we can show that the learning rates deduced earlier are asymptotically optimal under some circumstances.

**Corollary 2.** *Let $X \subset B_{\ell_2^d}$ be a domain such that we have an extension operator $\mathfrak{E}$ of the form described in front of Theorem 1. Furthermore, let $M > 0$, $Y := [-M, M]$, and $\mathrm{P}$ be a distribution on $X \times Y$ such that $\mathrm{P}_X$ is the uniform distribution on $X$. If, for some $\alpha \in \mathbb{N}$, we have $f_{L,\mathrm{P}}^* \in W_2^\alpha(\mathrm{P}_X)$, then, for all $\rho > 0$, both the SVM considered in Corollary 1 and the TV-SVM considered in Theorem 2 learn with the rate*

$$n^{-\frac{2\alpha}{2\alpha+d}+\rho}$$

*with probability $\mathrm{P}^n$ not less than $1 - e^{-\tau}$. Moreover, if $\alpha > d/2$, then this rate is asymptotically optimal in a minmax sense.*

Similar to Corollary 2 we can show assumption (6) and asymptotically optimal learning rates if the regression function is contained in a Besov space.

**Corollary 3.** *Let $X \subset B_{\ell_2^d}$ be a domain such that we have an extension operator $\mathfrak{E}$ of the form described in front of Theorem 1. Furthermore, let $M > 0$, $Y := [-M, M]$, and $\mathrm{P}$ be a distribution on $X \times Y$ such that $\mathrm{P}_X$ is the uniform distribution on $X$. If, for some $\alpha \geq 1$, we have $f_{L,\mathrm{P}}^* \in B_{2,\infty}^\alpha(\mathrm{P}_X)$, then, for all $\rho > 0$, both the SVM considered in Corollary 1 and the TV-SVM considered in Theorem 2 learn with the rate*

$$n^{-\frac{2\alpha}{2\alpha+d}+\rho}$$

*with probability $\mathrm{P}^n$ not less than $1 - e^{-\tau}$.*

Since for the entropy numbers $e_i(\mathrm{id} : B_{2,\infty}^\alpha(\mathrm{P}_X) \to L_2(\mathrm{P}_X)) \sim i^{-\frac{\alpha}{d}}$ holds (cf. [7, p. 151]) and since $B_{2,\infty}^\alpha(\mathrm{P}_X) = B_{2,\infty}^\alpha(X)$ is continuously embedded into the space $\ell_\infty(X)$ of all bounded functions on $X$, we obtain by [11, Theorem 2.2] that $n^{-\frac{2\alpha}{2\alpha+d}}$ is the optimal learning rate in a minmax sense for $\alpha > d$ (cf. [12, Theorem 13]). Therefore, for $\alpha > d$, the learning rates obtained in Corollary 3 are asymptotically optimal.

So far, we always assumed that $\mathrm{P}_X$ is the uniform distribution on $X$. This can be generalized by assuming that $\mathrm{P}_X$ is absolutely continuous w.r.t. the Lebesgue measure $\mu$ such that the corresponding density is bounded away from zero and from infinity. Then we have $L_2(\mathrm{P}_X) = L_2(\mu)$ with equivalent norms and the results for $\mu$ hold for $\mathrm{P}_X$ as well. Moreover, to derive learning rates, we actually only need that the Lebesgue density of $\mathrm{P}_X$ is upper bounded. The assumption that the density is bounded away from zero is only needed to derive the lower bounds in Corollaries 2 and 3.

Furthermore, we assumed $\gamma \in (0, 1]$ in Theorem 1, and hence in Corollary 1 and Theorem 2 as well. Note that $\gamma$ does not need to be restricted by one. Instead $\gamma$ only needs to be bounded from above by some constant such that estimates on the entropy numbers for Gaussian kernels as used in the proofs can be applied. For the sake of simplicity we have chosen one as upper bound, another upper bound would only have influence on the constants.

There have already been made several investigations on learning rates for SVMs using the least squares loss, see e.g. [13, 14, 15, 16, 17] and the references therein. In particular, optimal rates have been established in [16], if $f_\mathrm{P}^* \in H$, and the eigenvalue behavior of the integral operator associated to $H$ is known. Moreover, if $f_\mathrm{P}^* \notin H$ [17] and [12] establish both learning rates of the form $n^{-\beta/(\beta+p)}$, where $\beta$ is a parameter describing the approximation properties of $H$ and $p$ is a parameter describing the eigenvalue decay. Furthermore, in the introduction of [17] it is mentioned that the assumption on the eigenvalues and eigenfunctions also hold for Gaussian kernels with fixed width, but this case as well as the more interesting case of Gaussian kernels with variable widths are not further investigated. In the first case, where Gaussian kernels with fixed width are considered, the approximation error behaves very badly as shown in [18] and fast rates cannot be expected as we discuss below. In the second case, where variable widths are considered as in our paper, it is crucial to carefully control the influence of $\gamma$ on all arising constants which unfortunately has not been worked out in [17], either. In [17] and [12], however, additional assumptions on the interplay between $H$ and $L_2(\mathrm{P}_X)$ are required, and [17] actually considers a different exponent in the regularization term of (2). On the other hand, [12] shows that the rate $n^{-\beta/(\beta+p)}$ is often asymptotically optimal in a minmax sense. In particular, the latter is the case for $H = W_2^m(X)$, $f \in W_2^s(X)$, and $s \in (d/2, m]$, that is, when using a Sobolev space as the underlying RKHS $H$,

then all target functions contained in a Sobolev of lower smoothness $s > d/2$ can be learned with the asymptotically optimal rate $n^{-\frac{2s}{2s+d}}$. Here we note that the condition $s > d/2$ ensures by Sobolev's embedding theorem that $W_2^s(X)$ consists of bounded functions, and hence $Y = [-M, M]$ does not impose an additional assumption on $f_{L,\mathrm{P}}^*$. If $s \in (0, d/2]$, then the results of [12] still yield the above mentioned rates, but we no longer know whether they are optimal in a minmax sense, since $Y = [-M, M]$ does impose an additional assumption. In addition, note that for Sobolev spaces this result, modulo an extra log factor, has already been proved by [1]. This result suggests that by using a $C^\infty$-kernel such as the Gaussian RBF kernel, one could actually learn the entire scale of Sobolev spaces with the above mentioned rates. Unfortunately, however, there are good reasons to believe that this is not the case. Indeed, [18] shows that for many analytic kernels the approximation error can only have polynomial decay if $f_{L,\mathrm{P}}^*$ is analytic, too. In particular, for Gaussian kernels with *fixed* width $\gamma$ and $f_{L,\mathrm{P}}^* \notin C^\infty$ the approximation error does not decay polynomially fast, see [18, Proposition 1.1.], and if $f_{L,\mathrm{P}}^* \in W_2^m(X)$, then, in general, the approximation error function only has a logarithmic decay. Since it seems rather unlikely that these poor approximation properties can be balanced by superior bounds on the estimation error, the above-mentioned results indicate that Gaussian kernels with *fixed* width may have a poor performance. This conjecture is backed-up by many empirical experience gained throughout the last decade. Beginning with [19], research has thus focused on the learning performance of SVMs with varying widths. The result that is probably the closest to ours is [20]. Although these authors actually consider binary classification using convex loss functions including the least squares loss, formulated it is relatively straightforward to translate their finding to our least squares regression scenario. The result is the learning rate $n^{-\frac{m}{m+2d+2}}$, again under the assumption $f_{L,\mathrm{P}}^* \in W_2^m(X)$ for some $m > 0$. Furthermore, [21] treats the case, where $X$ is isometrically embedded into a $t$-dimensional, connected and compact $C^\infty$-submanifold of $\mathbb{R}^d$. In this case, it turns out that the resulting learning rate does not depend on the dimension $d$, but on the intrinsic dimension $t$ of the data. Namely the authors show the rate $n^{-\frac{s}{8s+4t}}$ modulo a logarithmic factor, where $s \in (0, 1]$ and $f_{L,\mathrm{P}}^* \in \mathrm{Lip}\,(s)$. Another direction of research that can be applied to Gaussian kernels with varying widths are multi-kernel regularization schemes, see [22, 23, 24] for some results in this direction. For example, [22] establishes learning rates of the form $n^{-\frac{2m-d}{4(4m-d)}+\rho}$ whenever $f_{L,\mathrm{P}}^* \in W_2^m(X)$ for some $m \in (d/2, d/2 + 2)$, where again $\rho > 0$ can be chosen to be arbitrarily close to $0$. Clearly, all these results provide rates that are far from being optimal, so that it seems fair to say that our results represent a significant advance. Furthermore, we can conclude that, in terms of asymptotical minmax rates, multi-kernel approaches applied Gaussian RBFs *cannot* provide any significant improvement over a simple training/validation approach for determining the kernel width and the regularization parameter, since the latter already leads to rates that are optimal modulo an arbitrarily small $\rho$ in the exponent.

## 3    Proof of the main result

To prove Theorem 1 we deduce an oracle inequality for the least squares loss by specializing [2, Theorem 7.23] (cf. Theorem 3). To be finally able to show Theorem 1 originating from Theorem 3, we have to estimate the approximation error.

**Lemma 1.** *Let $X \subset \mathbb{R}^d$ be a domain such that we have an extension operator $\mathfrak{E}$ of the form described in front of Theorem 1, $\mathrm{P}_X$ be the uniform distribution on $X$ and $f \in L_\infty(X)$. Furthermore, let $\tilde{f}$ be defined by*

$$\tilde{f}(x) := \left(\gamma\sqrt{\pi}\right)^{-\frac{d}{2}} \mathfrak{E}f(x) \tag{8}$$

*for all $x \in \mathbb{R}^d$ and, for $r \in \mathbb{N}$ and $\gamma > 0$, $K : \mathbb{R}^d \to \mathbb{R}$ be defined by*

$$K(\cdot) := \sum_{j=1}^{r} \binom{r}{j} (-1)^{1-j} \frac{1}{j^d} \left(\frac{2}{\gamma\sqrt{\pi}}\right)^{\frac{d}{2}} K_{\frac{j\gamma}{\sqrt{2}}}(\cdot) \tag{9}$$

*with*

$$K_\gamma(\cdot) := \exp\left(-\frac{\|\cdot\|_2^2}{\gamma^2}\right).$$

*Then, for $r \in \mathbb{N}$, $\gamma > 0$, and $q \in [1, \infty)$, we have $\mathfrak{E} f \in L_q(\widetilde{\mathrm{P}}_X)$ and*

$$\left\| K * \tilde{f} - f \right\|_{L_q(\mathrm{P}_X)}^q \leq C_{r,q}\, \omega_{r,L_q(\mathbb{R}^d)}^q\, (\mathfrak{E} f, \gamma/2) \ ,$$

*where $C_{r,q}$ is a constant only depending on $r$, $q$ and $\mu(X)$.*

In order to use the conclusion of Lemma 1 in the proof of Theorem 1 it is necessary to know some properties of $K * \tilde{f}$. Therefore, we need the next two lemmata.

**Lemma 2.** *Let $g \in L_2\left(\mathbb{R}^d\right)$, $H_\gamma$ be the RKHS of the Gaussian RBF kernel $k_\gamma$ over $X \subset \mathbb{R}^d$ and*

$$K\left(x\right) := \sum_{j=1}^{r} \binom{r}{j} (-1)^{1-j} \frac{1}{j^d} \left( \frac{2}{\gamma\sqrt{\pi}} \right)^{\frac{d}{2}} \exp\left( -\frac{2\left\| x \right\|_2^2}{j^2\gamma^2} \right)$$

*for $x \in \mathbb{R}^d$ and a fixed $r \in \mathbb{N}$. Then we have*

$$K * g \in H_\gamma \ ,$$
$$\left\| K * g \right\|_{H_\gamma} \leq (2^r - 1) \left\| g \right\|_{L_2(\mathbb{R}^d)} \ .$$

**Lemma 3.** *Let $g \in L_\infty\left(\mathbb{R}^d\right)$, $H_\gamma$ be the RKHS of the Gaussian RBF kernel $k_\gamma$ over $X \subset \mathbb{R}^d$ and $K$ be as in Lemma 2. Then*

$$\left| K * g\left(x\right) \right| \leq \left( \gamma\sqrt{\pi} \right)^{\frac{d}{2}} (2^r - 1) \left\| g \right\|_{L_\infty(\mathbb{R}^d)}$$

*holds for all $x \in X$. Additionally, we assume that $X$ is a domain in $\mathbb{R}^d$ such that we have an extension operator $\mathfrak{E}$ of the form described in front of Theorem 1, $Y := [-M, M]$ and, for all $x \in \mathbb{R}^d$, $\tilde{f}\left(x\right) := (\gamma\sqrt{\pi})^{-\frac{d}{2}} \mathfrak{E}\left(f_{L,\mathrm{P}}^*\left(x\right)\right)$, where $f_{L,\mathrm{P}}^*$ denotes a version of the conditional expectation such that $f_{L,\mathrm{P}}^*\left(x\right) = \mathbb{E}_\mathrm{P}\left(Y|x\right) \in [-M, M]$ for all $x \in X$. Then we have $\tilde{f} \in L_\infty\left(\mathbb{R}^d\right)$ and*

$$\left| K * \tilde{f}\left(x\right) \right| \leq a_{0,\infty} (2^r - 1) M$$

*for all $x \in X$, which implies*

$$L(y, K * \tilde{f}\left(x\right)) \leq 4^r a^2 M^2$$

*for the least squares loss $L$ and all $(x, y) \in X \times Y$.*

Next, we modify [2, Theorem 7.23], so that the proof of Theorem 1 can be build upon it.

**Theorem 3.** *Let $X \subset B_{\ell_2^d}$, $Y := [-M, M] \subset \mathbb{R}$ be a closed subset with $M > 0$ and $\mathrm{P}$ be a distribution on $X \times Y$. Furthermore, let $L : Y \times \mathrm{R} \to [0, \infty)$ be the least squares loss, $k_\gamma$ be the Gaussian RBF kernel over $X$ with width $\gamma \in (0, 1]$ and $H_\gamma$ be the associated RKHS. Fix an $f_0 \in H_\gamma$ and a constant $B_0 \geq 4M^2$ such that $\left\| L \circ f_0 \right\|_\infty \leq B_0$. Then, for all fixed $\tau \geq 1$, $\lambda > 0$, $\varepsilon > 0$ and $p \in (0, 1)$, the SVM using $H_\gamma$ and $L$ satisfies*

$$\lambda \left\| f_{\mathrm{D},\lambda} \right\|_{H_\gamma}^2 + \mathcal{R}_{L,\mathrm{P}}\left( \widehat{f}_{\mathrm{D},\lambda} \right) - \mathcal{R}_{L,\mathrm{P}}^*$$

$$\leq 9 \left( \lambda \left\| f_0 \right\|_{H_\gamma}^2 + \mathcal{R}_{L,\mathrm{P}}\left( f_0 \right) - \mathcal{R}_{L,\mathrm{P}}^* \right) + C_{\varepsilon,p} \frac{\gamma^{-(1-p)(1+\varepsilon)d}}{\lambda^p n} + \frac{\left( 3456 M^2 + 15 B_0 \right) (\ln(3) + 1)\tau}{n}$$

*with probability $\mathrm{P}^n$ not less than $1 - e^{-\tau}$, where $C_{\varepsilon,p}$ is a constant only depending on $\varepsilon$, $p$ and $M$.*

With the previous results we are finally able to prove the oracle inequality declared by Theorem 1.

*Proof of Theorem 1.* First of all, we want to apply Theorem 3 for $f_0 := K * \tilde{f}$ with

$$K\left(x\right) := \sum_{j=1}^{r} \binom{r}{j} (-1)^{1-j} \frac{1}{j^d} \left( \frac{2}{\gamma\sqrt{\pi}} \right)^{\frac{d}{2}} \exp\left( -\frac{2\left\| x \right\|_2^2}{j^2\gamma^2} \right)$$

and

$$\tilde{f}(x) := \left(\gamma\sqrt{\pi}\right)^{-\frac{d}{2}} \mathfrak{E} f_{L,P}^*(x)$$

for all $x \in \mathbb{R}^d$. The choice $f_{L,P}^*(x) \in [-M, M]$ for all $x \in X$ implies $f_{L,P}^* \in L_2(X)$ and the latter together with $X \subset B_{\ell_2^d}$ and (5) yields

$$
\begin{aligned}
\|\tilde{f}\|_{L_2(\mathbb{R}^d)} &= \left(\gamma\sqrt{\pi}\right)^{-\frac{d}{2}} \|\mathfrak{E} f_{L,P}^*\|_{L_2(\mathbb{R}^d)} \\
&\leq \left(\gamma\sqrt{\pi}\right)^{-\frac{d}{2}} a_{0,2} \|f_{L,P}^*\|_{L_2(X)} \\
&\leq \left(\frac{2}{\gamma\sqrt{\pi}}\right)^{\frac{d}{2}} a_{0,2} M ,
\end{aligned}
\tag{10}
$$

i.e. $\tilde{f} \in L_2\left(\mathbb{R}^d\right)$. Because of this and Lemma 2

$$f_0 = K * \tilde{f} \in H_\gamma$$

is satisfied and with Lemma 3 we have

$$\|L \circ f_0\|_\infty = \sup_{(x,y)\in X\times Y} |L(y, f_0(x))| = \sup_{(x,y)\in X\times Y} \left| L\left(y, K * \tilde{f}(x)\right) \right| \leq 4^r a^2 M^2 =: B_0 .$$

Furthermore, (1) and Lemma 1 yield

$$
\begin{aligned}
\mathcal{R}_{L,P}(f_0) - \mathcal{R}_{L,P}^* &= \mathcal{R}_{L,P}\left(K * \tilde{f}\right) - \mathcal{R}_{L,P}^* \\
&= \left\| K * \tilde{f} - f_{L,P}^* \right\|_{L_2(P_X)}^2 \\
&\leq C_{r,2}\, \omega_{r,L_2(\mathbb{R}^d)}^2 \left( \mathfrak{E} f_{L,P}^*, \frac{\gamma}{2}\right) \\
&\leq C_{r,2}\, c^2 \gamma^{2\alpha} ,
\end{aligned}
$$

where we used the assumption

$$\omega_{r,L_2(\mathbb{R}^d)}\left(\mathfrak{E} f_{L,P}^*, \frac{\gamma}{2}\right) \leq c\gamma^\alpha$$

for $\gamma \in (0,1]$, $\alpha \geq 1$, $r = \lfloor\alpha\rfloor + 1$ and a constant $c > 0$ in the last step. By Lemma 2 and (10) we know

$$\|f_0\|_{H_\gamma} = \|K * \tilde{f}\|_{H_\gamma} \leq (2^r - 1) \|\tilde{f}\|_{L_2(\mathbb{R}^d)} \leq (2^r - 1) \left(\frac{2}{\gamma\sqrt{\pi}}\right)^{\frac{d}{2}} a_{0,2} M .$$

Therefore, Theorem 3 and the above choice of $f_0$ yield, for all fixed $\tau \geq 1$, $\lambda > 0$, $\varepsilon > 0$ and $p \in (0,1)$, that the SVM using $H_\gamma$ and $L$ satisfies

$$
\begin{aligned}
&\lambda \|f_{D,\lambda}\|_{H_\gamma}^2 + \mathcal{R}_{L,P}\left(\widehat{f}_{D,\lambda}\right) - \mathcal{R}_{L,P}^* \\
&\leq 9\left(\lambda (2^r - 1)^2 \left(\frac{2}{\gamma\sqrt{\pi}}\right)^d a_{0,2}^2 M^2 + C_{r,2} c^2 \gamma^{2\alpha}\right) \\
&\quad + C_{\varepsilon,p} \frac{\gamma^{-(1-p)(1+\varepsilon)d}}{\lambda^p n} + \frac{\left(3456 + 15 \cdot 4^r a^2\right) M^2 (\ln(3)+1)\tau}{n} \\
&\leq C_1 \lambda \gamma^{-d} + 9\, C_r c^2 \gamma^{2\alpha} + C_{\varepsilon,p} \frac{\gamma^{-(1-p)(1+\varepsilon)d}}{\lambda^p n} + \frac{C_2 \tau}{n}
\end{aligned}
$$

with probability $P^n$ not less than $1 - e^{-\tau}$ and with constants $C_1 := 9\,(2^r - 1)^2\, 2^d \pi^{-\frac{d}{2}} a_{0,2}^2 M^2$, $C_2 := (\ln(3)+1)\left(3456 + 15 \cdot 4^r a^2\right) M^2$, $a := \max\{a_{0,\infty}, 1\}$, $C_r := C_{r,2}$ only depending on $r$ and $\mu(X)$ and $C_{\varepsilon,p}$ as in Theorem 3. $\qquad\square$

# References

[1] L. Györfi, M. Kohler, A. Krzyżak, and H. Walk. *A Distribution-Free Theory of Nonparametric Regression*. Springer-Verlag New York, 2002.

[2] I. Steinwart and A. Christmann. *Support Vector Machines*. Springer-Verlag, New York, 2008.

[3] R.A. DeVore and G.G. Lorentz. *Constructive Approximation*. Springer-Verlag Berlin Heidelberg, 1993.

[4] R.A. DeVore and V.A. Popov. Interpolation of Besov Spaces. *AMS*, Volume 305, 1988.

[5] H. Berens and R.A. DeVore. Quantitative Korovin theorems for positive linear operators on $L_p$-spaces. *AMS*, Volume 245, 1978.

[6] H. Johnen and K. Scherer. On the equivalence of the K-functional and moduli of continuity and some applications. In *Lecture Notes in Math.*, volume 571, pages 119–140. Springer-Verlag Berlin, 1976.

[7] D.E. Edmunds and H. Triebel. *Function Spaces, Entropy Numbers, Differential 0perators*. Cambridge University Press, 1996.

[8] E.M. Stein. *Singular Integrals and Differentiability Properties of Functions*. Princeton Univ. Press, 1970.

[9] R.A. Adams and J.J.F. Fournier. *Sobolev Spaces*. Academic Press, $2^{nd}$ edition, 2003.

[10] H. Triebel. *Theory of Function Spaces III*. Birkhäuser Verlag, 2006.

[11] V. Temlyakov. Optimal estimators in learning theory. *Banach Center Publications, Inst. Math. Polish Academy of Sciences*, 72:341–366, 2006.

[12] I. Steinwart, D. Hush, and C. Scovel. Optimal rates for regularized least squares regression. *Proceedings of the 22nd Annual Conference on Learning Theory*, 2009.

[13] F. Cucker and S. Smale. On the mathematical foundations of learning. *Bull. Amer. Math. Soc.*, 39:1–49, 2002.

[14] E. De Vito, A. Caponnetto, and L. Rosasco. Model selection for regularized least-squares algorithm in learning theory. *Found. Comput. Math.*, 5:59–85, 2005.

[15] S. Smale and D.-X. Zhou. Learning theory estimates via integral operators and their approximations. *Constr. Approx.*, 26:153–172, 2007.

[16] A. Caponnetto and E. De Vito. Optimal rates for regularized least squares algorithm. *Found. Comput. Math.*, 7:331–368, 2007.

[17] S. Mendelson and J. Neeman. Regularization in kernel learning. *Ann. Statist.*, 38:526–565, 2010.

[18] S. Smale and D.-X. Zhou. Estimating the approximation error in learning theory. *Anal. Appl.*, Volume 1, 2003.

[19] I. Steinwart and C. Scovel. Fast rates for support vector machines using Gaussian kernels. *Ann. Statist.*, 35:575–607, 2007.

[20] D.-H. Xiang and D.-X. Zhou. Classification with Gaussians and convex loss. *J. Mach. Learn. Res.*, 10:1447–1468, 2009.

[21] G.-B. Ye and D.-X. Zhou. Learning and approximation by Gaussians on Riemannian manifolds. *Adv. Comput. Math.*, Volume 29, 2008.

[22] Y. Ying and D.-X. Zhou. Learnability of Gaussians with flexible variances. *J. Mach. Learn. Res. 8*, 2007.

[23] C.A. Micchelli, M. Pontil, Q. Wu, and D.-X. Zhou. Error bounds for learning the kernel. 2005.

[24] Y. Ying and C. Campbell. Generalization bounds for learning the kernel. In S. Dasgupta and A. Klivans, editors, *Proceedings of the 22nd Annual Conference on Learning Theory*, 2009.

